# Efficient Supervised Sparse Analysis and Synthesis Operators

**Pablo Sprechmann**
Duke University
pablo.sprechmann@duke.edu

**Roee Litman**
Tel Aviv University
roeelitman@post.tau.ac.il

**Tal Ben Yakar**
Tel Aviv University
talby10@gmail.com

**Alex Bronstein**
Tel Aviv University
bron@eng.tau.ac.il

**Guillermo Sapiro**
Duke University
guillermo.sapiro@duke.edu
*

## Abstract

In this paper, we propose a new computationally efficient framework for learning sparse models. We formulate a unified approach that contains as particular cases models promoting sparse synthesis and analysis type of priors, and mixtures thereof. The supervised training of the proposed model is formulated as a bilevel optimization problem, in which the operators are optimized to achieve the best possible performance on a specific task, e.g., reconstruction or classification. By restricting the operators to be shift invariant, our approach can be thought as a way of learning sparsity-promoting convolutional operators. Leveraging recent ideas on fast trainable regressors designed to approximate exact sparse codes, we propose a way of constructing feed-forward networks capable of approximating the learned models at a fraction of the computational cost of exact solvers. In the shift-invariant case, this leads to a principled way of constructing a form of task-specific convolutional networks. We illustrate the proposed models on several experiments in music analysis and image processing applications.

## 1 Introduction

Parsimony, preferring a simple explanation to a more complex one, is probably one of the most intuitive principles widely adopted in the modeling of nature. The past two decades of research have shown the power of parsimonious representation in a vast variety of applications from diverse domains of science. Parsimony in the form of *sparsity* has been shown particularly useful in the fields of signal and image processing and machine learning. Sparse models impose sparsity-promoting priors on the signal, which can be roughly categorized as *synthesis* or *analysis*. Synthesis priors are generative, asserting that the signal is approximated well as a superposition of a small number of vectors from a (possibly redundant) synthesis dictionary. Analysis priors, on the other hand, assume that the signal admits a sparse projection onto an analysis dictionary. Many classes of signals, in particular, speech, music, and natural images, have been shown to be sparsely representable in overcomplete wavelet and Gabor frames, which have been successfully adopted as synthesis dictionaries in numerous applications [14]. Analysis priors involving differential operators, of which total variation is a popular instance, have also been shown very successful in regularizing ill-posed image restoration problems [19].

Despite the spectacular success of these axiomatically constructed synthesis and analysis operators, significant empirical evidence suggests that better performance is achieved when a data- or problem-specific dictionary is used instead of a predefined one. Works [1, 16], followed by many others, demonstrated that synthesis dictionaries can be constructed to best represent training data by solving essentially a matrix factorization problem. Despite the lack of convexity, many efficient dictionary learning procedures have been proposed.

This *unsupervised* or data-driven approach to synthesis dictionary learning is well-suited for reconstruction tasks such as image restoration. For example, synthesis models with learned dictionaries, have achieved excellent results in denoising [9, 13]. However, in discriminative tasks such as classification, good data reconstruction is not necessarily required or even desirable. Attempts to replicate the success of sparse models in discriminative tasks led to the recent interest in *supervised* or a task- rather than data-driven dictionary learning, which appeared to be a significantly more difficult modeling and computational problem compared to its unsupervised counterpart [6].

Supervised learning also seems to be the only practical option for learning unstructured non-generative analysis operators, for which no simple unsupervised alternatives exist. While the supervised analysis operator learning has been mainly used as regularization on inverse problems, e.g., denoising [5], we argue that it is often better suited for classification tasks than it synthesis counterpart, since the feature learning and the reconstruction are separated. Recent works proposed to address the supervised learning of $\ell_1$ norm synthesis [12] and analysis [5, 17] priors via bilevel optimization [8], in which the minimization of a task-specific loss with respect to a dictionary depends in turn on the minimizer of a representation pursuit problem using that dictionary.

For the synthesis case, the task-oriented bilevel optimization problem is smooth and can be efficiently solved using stochastic gradient descent (SGD) [12]. However, [12] heavily relies on the separability of the proximal operator of the $\ell_1$ norm, and thus cannot be extended to the analysis case, where the $\ell_1$ term is not separable. The approach proposed in [17] formulates an analysis model with a smoothed $\ell_1$-type prior and uses implicit differentiation to obtain its gradients with respect to the dictionary required for the solution of the bilevel problem. However, such approximate priors are known to produce inferior results compared to their exact counterparts.

**Main contributions.** This paper focuses on supervised learning of synthesis and analysis priors, making three main contributions:

First, we consider a more general sparse model encompassing analysis and synthesis priors as particular cases, and formulate its supervised learning as a bilevel optimization problem. We propose a new analysis technique, for which the (almost everywhere) smoothness of the proposed bilevel problem is shown, and its exact subgradients are derived. We also show that the model can be extended to include a sensing matrix and a non-Euclidean metric in the data term, both of which can be learned as well. We relate the learning of the latter metric matrix to task-driven metric learning techniques.

Second, we show a systematic way of constructing fast fixed-complexity approximations to the solution of the proposed exact pursuit problem by unrolling few iterations of the exact iterative solver into a feed-forward network, whose parameters are learned in the supervised regime. The idea of deriving a fast approximation of sparse codes from an iterative algorithm has been recently successfully advocated in [11] for the synthesis model. We present an extension of this line of research to the various settings of analysis-flavored sparse models.

Third, we dedicate special attention to the shift-invariant particular case of our model. The fast approximation in this case assumes the form of a convolutional neural network.

## 2  Analysis, synthesis, and mixed sparse models

We consider a generalization of the Lasso-type [21, 22] pursuit problem

$$\min_{\mathbf{y}} \frac{1}{2}\|\mathbf{M}_1\mathbf{x} - \mathbf{M}_2\mathbf{y}\|_2^2 + \lambda_1\|\mathbf{\Omega}\mathbf{y}\|_1 + \frac{\lambda_2}{2}\|\mathbf{y}\|_2^2, \tag{1}$$

where $\mathbf{x} \in \mathbb{R}^n$, $\mathbf{y} \in \mathbb{R}^k$, $\mathbf{M}_1$, $\mathbf{M}_2$ are $m \times n$ and $m \times k$, respectively, $\mathbf{\Omega}$ is $r \times k$, and $\lambda_1, \lambda_2 > 0$ are parameters. Pursuit problem (1) encompasses many important particular cases that have been extensively studied in the literature: By setting $\mathbf{M}_1 = \mathbf{I}$, $\mathbf{\Omega} = \mathbf{I}$, and $\mathbf{M}_2 = \mathbf{D}$ to be a column-overcomplete dictionary ($k > m$), the standard sparse synthesis model is obtained, which attempts to

**input** : Data $\mathbf{x}$, matrices $\mathbf{M}_1, \mathbf{M}_2, \mathbf{\Omega}$, weights $\lambda_1, \lambda_2$, parameter $\rho > 0$.
**output**: Sparse code $\mathbf{y}$.
Initialize $\boldsymbol{\mu}^0 = \mathbf{0}$, $\mathbf{z}^0 = \mathbf{0}$
**for** $j = 1, 2, \ldots$ *until convergence* **do**
$\quad\quad \mathbf{y}^{j+1} = (\mathbf{M}_2^{\mathrm{T}} \mathbf{M}_2 + \rho \mathbf{\Omega}^{\mathrm{T}} \mathbf{\Omega} + \lambda_2 \mathbf{I})^{-1} (\mathbf{M}_2^{\mathrm{T}} \mathbf{M}_1 \mathbf{x} + \rho \mathbf{\Omega}^{\mathrm{T}} (\mathbf{z}^j - \boldsymbol{\mu}^j))$
$\quad\quad \mathbf{z}^{j+1} = \sigma_{\frac{\lambda_1}{\rho}} (\mathbf{\Omega} \mathbf{y}^{j+1} + \boldsymbol{\mu}^j)$
$\quad\quad \boldsymbol{\mu}^{j+1} = \boldsymbol{\mu}^j + \mathbf{\Omega} \mathbf{y}^{j+1} - \mathbf{z}^{j+1}$
**end**
**Algorithm 1:** Alternating direction method of multipliers (ADMM). Here, $\sigma_t(z) = \mathrm{sign}(z) \cdot \max\{|z| - t, 0\}$ denotes the element-wise soft thresholding (the proximal operator of $\ell_1$).

represent the data vector $\mathbf{x}$ as a sparse linear combination of the atoms of $\mathbf{D}$. The case where the data are unavailable directly, but rather through a set of (usually fewer, $m < n$) linear measurements, is handled by supplying $\mathbf{x} \in \mathbb{R}^m$ and setting $\mathbf{M}_2 = \mathbf{\Phi} \mathbf{D}$, with $\mathbf{\Phi}$ being an $m \times n$ sensing matrix. Such a case arises frequently in compressed sensing applications as well as in general inverse problems.

One the other hand, by setting $\mathbf{M}_1, \mathbf{M}_2 = \mathbf{I}$, and $\mathbf{\Omega}$ a row-overcomplete dictionary ($r > k$), the standard sparse analysis model is obtained, which attempts to approximate the data vector $\mathbf{x}$ by another vector $\mathbf{y}$ in the same space admitting a sparse projection on $\mathbf{\Omega}$. For example, by setting $\mathbf{\Omega}$ to be the matrix of discrete derivatives leads to total variation regularization, which has been shown extremely successful in numerous signal processing applications. The analysis model can also be extended by adding an $m \times k$ sensing operator $\mathbf{M}_2 = \mathbf{\Phi}$, assuming that $\mathbf{x}$ is given in the $m$-dimensional measurement space. This leads to popular analysis formulations of image deblurring, super-resolution, and other inverse problems.

Keeping both the analysis and the synthesis dictionaries and setting $\mathbf{M}_2 = \mathbf{D}$, $\mathbf{\Omega} = [\mathbf{\Omega}'\mathbf{D}; \mathbf{I}]$, leads to the *mixed* model. Note that the reconstructed data vector is now obtained by $\hat{\mathbf{x}} = \mathbf{D}\mathbf{y}$ with sparse $\mathbf{y}$; as a result, the $\ell_1$ term is extended to make sparse the projection of $\hat{\mathbf{x}}$ on the analysis dictionary $\mathbf{\Omega}'$, as well as impose sparsity of $\mathbf{y}$. A sensing matrix can be incorporated in this setting as well, by setting $\mathbf{M}_1 = \mathbf{\Phi}$ and $\mathbf{M}_2 = \mathbf{\Phi}\mathbf{D}$. Alternatively, we can interpret $\mathbf{\Phi}$ as the projection matrix parametrizing a $\mathbf{\Phi}^{\mathrm{T}}\mathbf{\Phi}$ Mahalanobis metric, thus generalizing the traditional Euclidean data term.

A particularly important family of analysis operators is obtained when the operator is restricted to be shift-invariant. In this case, the operator can be expressed as a convolution with a filter, $\boldsymbol{\gamma} * \mathbf{y}$, whose impulse response $\boldsymbol{\gamma} \in \mathbb{R}^f$ is generally of a much smaller dimension than $\mathbf{y}$. A straightforward generalization would be to consider an analysis operator consisting of $q$ filters,

$$\mathbf{\Omega}(\boldsymbol{\gamma}_1, \ldots, \boldsymbol{\gamma}_q) = \left[ \mathbf{\Omega}_1(\boldsymbol{\gamma}_1); \cdots ; \mathbf{\Omega}_q(\boldsymbol{\gamma}_q) \right] \qquad \text{with} \qquad \mathbf{\Omega}_i \mathbf{y} = \boldsymbol{\gamma}_i * \mathbf{y}, \qquad 1 \le i \le q. \quad (2)$$

This model includes as a particular case the isotropic total variation priors. In this case, $q = 2$ and the filters correspond to the discrete horizontal and vertical derivatives. In general, the exact form of the operator depends on the dimension of the convolution, and the type of boundary conditions.

On of the most attractive properties of pursuit problem (1) is convexity, which becomes strict for $\lambda_2 > 0$. While for $\mathbf{\Omega} = \mathbf{I}$, (1) can be solved efficiently using the popular proximal methods [15] (such as FISTA [2]), this is no more an option in the case of a non-trivial $\mathbf{\Omega}$, as $\|\mathbf{\Omega}\mathbf{y}\|_1$ has no more a closed-form proximal operator. A way to circumvent this difficulty is by introducing an auxiliary variable $\mathbf{z} = \mathbf{\Omega}\mathbf{y}$ and solving the constrained convex program

$$\min_{\mathbf{y}, \mathbf{z}} \frac{1}{2} \|\mathbf{M}_1 \mathbf{x} - \mathbf{M}_2 \mathbf{y}\|_2^2 + \lambda_1 \|\mathbf{z}\|_1 + \frac{\lambda_2}{2} \|\mathbf{y}\|_2^2 \quad \text{s.t} \quad \mathbf{z} = \mathbf{\Omega}\mathbf{y}, \quad (3)$$

with an unscaled $\ell_1$ term. This leads to a family of the so-called split-Bregman methods; the application of augmented Lagrangian techniques to solve (3) is known in the literature as *alternating direction method of multipliers* (ADMM) [4], summarized in Algorithm 1. Particular instances might be solved more efficiently with alternative algorithms (i.e. proximal splitting methods).

## 3  Bilevel sparse models

A central focus of this paper is to develop a framework for supervised learning of the parameters in (1), collectively denoted by $\Theta = \{\mathbf{M}_1, \mathbf{M}_2, \mathbf{D}, \mathbf{\Omega}\}$, to achieve the best possible performance in a

specific task such as reconstruction or classification. Supervised schemes arise very naturally when dealing with analysis operators. In sharp contrast to the generative synthesis models, where data reconstruction can be enforced unsupervisedly, there is no trivial way for unsupervised training of analysis operators without restricting them to satisfy some external, frequently arbitrary, constraints. Clearly, unconstrained minimization of (1) over $\boldsymbol{\Omega}$ would lead to a trivial solution $\boldsymbol{\Omega} = \mathbf{0}$. The ideas proposed in [12] fit very well here, and were in fact used in [5, 17] for learning of unstructured analysis operators. However, in both cases the authors used a smoothed version of the $\ell_1$ penalty, which is known to produce inferior results. In this work we extend these ideas, without smoothing the penalty. Formally, given an observed variable $\mathbf{x} \in \mathbb{R}^n$ coming from a certain distribution $P_{\mathcal{X}}$, we aim at predicting a corresponding latent variable $\mathbf{y} \in \mathbb{R}^k$. The latter can be discrete, representing a label in a classification task, or continuous like in regression or reconstruction problems. As noted before, when $\lambda_2 > 0$, problem (1) is strictly convex and, consequently, has a unique minimizer. The solution of the pursuit problem defines, therefore, an unambiguous deterministic map from the space of the observations to the space of the latent variables, which we denote by $\mathbf{y}_{\Theta}^*(\mathbf{x})$. The map depends on the model parameters $\Theta$. The goal of supervised learning is to select such $\Theta$ that minimize the expectation over $P_{\mathcal{X}}$ of some problem-specific loss function $\ell$. In practice, the distribution $P_{\mathcal{X}}$ is usually unknown, and the expected loss is substituted by an empirical loss computed on a training set of pairs $(\mathbf{x}, \mathbf{y}) \in (\mathcal{X}, \mathcal{Y})$. The task-driven model learning problem becomes [12]

$$\min_{\Theta} \frac{1}{|\mathcal{X}|} \sum_{(\mathbf{x}, \mathbf{y}) \in (\mathcal{X}, \mathcal{Y})} \ell(\mathbf{y}, \mathbf{x}, \mathbf{y}_{\Theta}^*(\mathbf{x})) + \phi(\Theta), \tag{4}$$

where $\phi(\Theta)$ denotes a regularizer on the model parameters added to stabilize the solution. Problem (4) is a bilevel optimization problem [8], as we need to optimize the loss function $\ell$, which in turn depends on the minimizer of (1).

As an example, let us examine the generic class of signal reconstruction problems, in which, as explained in Section 2, the matrix $\mathbf{M}_2 = \boldsymbol{\Phi}$ plays the role of a linear degradation (e.g., blur and sub-sampling in case of image super-resolution problems), producing the degraded and, possibly, noisy observation $\mathbf{x} = \boldsymbol{\Phi}\mathbf{y} + \mathbf{n}$ from the latent clean signal $\mathbf{y}$. The goal of the model learning problem is to select the model parameters $\Theta$ yielding the most accurate inverse operator, $\mathbf{y}_{\Theta}^*(\boldsymbol{\Phi}\mathbf{y}) \approx \mathbf{y}$. Assuming a simple white Gaussian noise model, this can be achieved through the following loss

$$\ell(\mathbf{y}, \mathbf{x}, \mathbf{y}^*) \quad = \quad \frac{1}{2}\|\mathbf{y} - \mathbf{y}^*\|_2^2. \tag{5}$$

While the supervised learning of analysis operator has been considered for solving denoising problems [5, 17], here we address more general scenarios. In particular, we argue that, when used along with metric learning, it is often better suited for classification tasks than its synthesis counterpart, because the non-generative nature of analysis models is more suitable for feature learning. For simplicity, we consider the case of a linear binary classifier of the form $\mathrm{sign}(\mathbf{w}^{\mathrm{T}}\mathbf{z} + b)$ operating on the "feature vector" $\mathbf{z} = \boldsymbol{\Omega}\mathbf{y}_{\Theta}^*(\mathbf{x})$. Using a loss of the form $\ell(y, \mathbf{x}, \mathbf{z}) = f(-y(\mathbf{w}^{\mathrm{T}}\mathbf{z} + b))$, with $f$ being, e.g., the logistic regression function $f(t) = \log(1 + e^{-t})$, we train the model parameters $\Theta$ simultaneously with the classifier parameters $\mathbf{w}, b$. In this context, the learning of $\Theta$ can be interpreted as feature learning.

The generalization to multi-class classification problems is straightforward, by using a matrix $\mathbf{W}$ and a vector $\mathbf{b}$ instead of $\mathbf{w}$ and $b$. It is worthwhile noting that more stable classifiers are obtained by adding a regularization of the form $\phi = \|\mathbf{W}\|_{\mathrm{F}}^2$ to the learning problem (4).

**Optimization.** A local minimizer of the non-convex model learning problem (4) can be found via stochastic optimization [8, 12, 17], by performing gradient descent steps on each of the variables in $\Theta$ with the pair $(\mathbf{x}, \mathbf{y})$ each time drawn at random from the training set. Specifically, the parameters at iteration $i + 1$ are obtained by

$$\Theta^{i+1} \leftarrow \Theta^i - \eta_i \nabla_{\Theta} \ell(\mathbf{x}, \mathbf{y}, \mathbf{y}_{\Theta^i}^*(\mathbf{x})), \tag{6}$$

where $0 \leq \eta_i \leq \eta$ is a decreasing sequence of step-sizes. Following [12], we use a step size of the form $\eta_i = \min(\eta, \eta i_0/i)$ in all our experiments, which means that a fixed step size is used during the first $k_0$ iterations, after which it decays according to the $1/i$ annealing strategy. Note that the learning requires the gradient $\nabla_{\Theta}\ell$, which in turn relies on the gradient of $\mathbf{y}_{\Theta}^*(\mathbf{x})$ with respect to $\Theta$. Even though $\mathbf{y}_{\Theta}^*(\mathbf{x})$ is obtained by solving a non-smooth optimization problem, we will

show that it is almost everywhere differentiable, and one can compute its gradient with respect to $\Theta = \{\mathbf{M}_1, \mathbf{M}_2, \mathbf{D}, \mathbf{\Omega}\}$ explicitly and in closed form. In the next section, we briefly summarize the derivation of the gradients for $\nabla_{\mathbf{M}_2}\ell$ and $\nabla_{\mathbf{\Omega}}\ell$, as these two are the most interesting cases. The gradients needed for the remaining model settings described in Section 2 can be obtained straightforwardly from $\nabla_{\mathbf{M}_2}\ell$ and $\nabla_{\mathbf{\Omega}}\ell$.

**Gradient computation.** To obtain the gradients of the cost function with respect to the matrices $\mathbf{M}_2$ and $\mathbf{\Omega}$, we consider a version of (3) in which the equality constrained is relaxed by a penalty,

$$\min_{\mathbf{z},\mathbf{y}} \frac{1}{2}\|\mathbf{M}_1\mathbf{x} - \mathbf{M}_2\mathbf{y}\|_2^2 + \frac{t}{2}\|\mathbf{\Omega}\mathbf{y} - \mathbf{z}\|_2^2 + \lambda_1\|\mathbf{z}\|_1 + \frac{\lambda_2}{2}\|\mathbf{y}\|_2^2, \qquad (7)$$

with $t > 0$ being the penalty parameter. We denote by $\mathbf{y}_t^*$ and $\mathbf{z}_t^*$ the unique minimizers of this strongly convex optimization problem with $t$, $\mathbf{x}$, $\mathbf{M}_1$, $\mathbf{M}_2$ and $\mathbf{\Omega}$ fixed. Naturally, $\mathbf{y}_t^*$ and $\mathbf{z}_t^*$ are functions of $\mathbf{x}$ and $\Theta$, the same way as $\mathbf{y}_\Theta^*(\mathbf{x})$. Throughout this section, we will omit this dependence to simplify notation. The first-order optimality conditions of (8) lead to the equalities

$$\mathbf{M}_2^{\mathrm{T}}(\mathbf{M}_2\mathbf{y}_t^* - \mathbf{M}_1\mathbf{x}) + t\mathbf{\Omega}^{\mathrm{T}}(\mathbf{\Omega}\mathbf{y}_t^* - \mathbf{z}_t^*) + \lambda_2\mathbf{y}_t^* = 0, \qquad (8)$$
$$t(\mathbf{z}_t^* - \mathbf{\Omega}\mathbf{y}_t^*) + \lambda_1(\mathrm{sign}(\mathbf{z}_t^*) + \boldsymbol{\alpha}) = 0, \qquad (9)$$

where the sign of zero is defined as zero and $\boldsymbol{\alpha}$ is a vector in $\mathbb{R}^r$ such that $\boldsymbol{\alpha}_\Lambda = 0$ and $|\boldsymbol{\alpha}_{\Lambda^c}| \leq 1$. Here, $\boldsymbol{\alpha}_\Lambda$ denotes the sub-vector of $\boldsymbol{\alpha}$ whose rows are reduced to $\Lambda$, the set of non-zero coefficients (active set) of $\mathbf{z}_t^*$.

It has been shown that the solution of the synthesis [12], analysis [23], and generalized Lasso [22] regularization problems are all piecewise affine functions of the observations and the regularization parameter. This means that the active set of the solution is constant on intervals of the regularization parameter $\lambda_1$. Moreover, the number of transition points (values of $\lambda_1$ that for a given observation $\mathbf{x}$ the active set of the solution changes) is finite and thus negligible. It can be shown that if $\lambda_1$ is not a transition point of $\mathbf{x}$, then a small perturbation in $\mathbf{\Omega}$, $\mathbf{M}_1$, or $\mathbf{M}_2$ leaves $\Lambda$ and the sign of the coefficients in the solution unchanged [12]. Applying this result to (8), we can state that $\mathrm{sign}(\mathbf{z}_t^*) = \mathrm{sign}(\mathbf{\Omega}\mathbf{y}_t^*)$.

Let $\mathbf{I}_\Lambda$ be the projection onto $\Lambda$, and let $\mathbf{P}_\Lambda = \mathbf{I}_\Lambda^{\mathrm{T}}\mathbf{I}_\Lambda = \mathrm{diag}\{|\mathrm{sign}(\mathbf{z}^*)|\}$ denote the matrix setting to zero the rows corresponding to $\Lambda^c$. Multiplying the second optimality condition by $\mathbf{P}_\Lambda$, we have $\mathbf{z}_t^* = \mathbf{P}_\Lambda\mathbf{z}_t^* = \mathbf{P}_\Lambda\mathbf{\Omega}\mathbf{y}_t^* - \frac{\lambda_1}{t}\mathrm{sign}(\mathbf{z}_t^*)$, where we used the fact that $\mathbf{P}_\Lambda\mathrm{sign}(\mathbf{z}_t^*) = \mathrm{sign}(\mathbf{z}_t^*)$. We can plug the latter result into (9), obtaining

$$\mathbf{y}_t^* = \mathbf{Q}_t(\mathbf{M}_2^{\mathrm{T}}\mathbf{M}_1\mathbf{x} - \lambda_1\mathbf{\Omega}^{\mathrm{T}}\mathrm{sign}(\mathbf{z}_t^*)), \qquad (10)$$

where $\mathbf{Q}_t = (t\mathbf{\Omega}^{\mathrm{T}}\mathbf{P}_{\Lambda^c}\mathbf{\Omega} + \mathbf{B})^{-1}$ and $\mathbf{B} = \mathbf{M}_2^{\mathrm{T}}\mathbf{M}_2 + \lambda_2\mathbf{I}$. By using the first-order Taylor's expansion of (11), we can obtain an expression for the gradients of $\ell(\mathbf{y}_t^*)$ with respect to $\mathbf{M}_2$ and $\mathbf{\Omega}$,

$$\nabla_{\mathbf{\Omega}}\ell(\mathbf{y}_t^*) = -\lambda_1\mathrm{sign}(\mathbf{z}_t^*)\boldsymbol{\beta}^{\mathrm{T}} - \mathbf{P}_{\Lambda^c}\mathbf{\Omega}(t\mathbf{y}_t^*\boldsymbol{\beta}_t^{\mathrm{T}} + t\boldsymbol{\beta}_t\mathbf{y}_t^{*\mathrm{T}}), \qquad (11)$$
$$\nabla_{\mathbf{M_2}}\ell(\mathbf{y}_t^*) = \mathbf{M_2}(\mathbf{y}_t^*\boldsymbol{\beta}_t^{\mathrm{T}} + \boldsymbol{\beta}_t\mathbf{y}_t^{*\mathrm{T}}), \qquad (12)$$

where $\boldsymbol{\beta}_t = \mathbf{Q}_t\nabla_{\mathbf{y}^*}\ell(\mathbf{y}_t^*)$.

Note that since the (unique) solution of (8) can be made arbitrarily close to the (unique) solution of (1) by increasing $t$, we can obtain the exact gradients of $\mathbf{y}^*$ by taking the limit $t \to \infty$ in the above expressions. First, observe that

$$\mathbf{Q}_t = (t\mathbf{\Omega}^{\mathrm{T}}\mathbf{P}_{\Lambda^c}\mathbf{\Omega} + \mathbf{B})^{-1} = (\mathbf{B}(t\mathbf{B}^{-1}\mathbf{\Omega}^{\mathrm{T}}\mathbf{P}_{\Lambda^c}\mathbf{\Omega} + \mathbf{I}))^{-1} = (t\mathbf{C} + \mathbf{I})^{-1}\mathbf{B}^{-1},$$

where $\mathbf{C} = \mathbf{B}^{-1}\mathbf{\Omega}^{\mathrm{T}}\mathbf{P}_{\Lambda^c}\mathbf{\Omega}$. Note that $\mathbf{B}$ is invertible if $\mathbf{M}_2$ is full-rank or if $\lambda_2 > 0$. Let $\mathbf{C} = \mathbf{U}\mathbf{H}\mathbf{U}^{-1}$ be the eigen-decomposition of $\mathbf{C}$, with $\mathbf{H}$ a diagonal matrix with the elements $h_i$, $1 \leq i \leq n$. Then, $\mathbf{Q}_t = \mathbf{U}\mathbf{H}_t\mathbf{U}^{-1}\mathbf{B}^{-1}$, where $\mathbf{H}_t$ is diagonal with $1/(th_i + 1)$ on the diagonal. In the limit, $th_i \to 0$ if $h_i = 0$, and $th_i \to \infty$ otherwise, yielding

$$\mathbf{Q} = \lim_{t\to\infty}\mathbf{Q}_t = \mathbf{U}\mathbf{H}'\mathbf{U}^{-1}\mathbf{B}^{-1} \quad \text{with } \mathbf{H}' = \mathrm{diag}\{h_i'\}, \quad h_i' = \begin{cases} 0 & : h_i \neq 0, \\ 1 & : h_i = 0. \end{cases} \qquad (13)$$

The optimum of (1) is given by $\mathbf{y}^* = \mathbf{Q}(\mathbf{M}_2^{\mathrm{T}}\mathbf{M}_1\mathbf{x} - \lambda_1\mathbf{\Omega}^{\mathrm{T}}\mathrm{sign}(\mathbf{z}^*))$. Analogously, we take the limit in the expressions describing the gradients in (12) and (13). We summarize our main result in Proposition 1 below, for which we define

$$\tilde{\mathbf{Q}} = \lim_{t\to\infty}t\mathbf{Q}_t = \mathbf{U}\mathbf{H}''\mathbf{U}^{-1}\mathbf{B}^{-1} \quad \text{with } \mathbf{H}'' = \mathrm{diag}\{h_i''\}, \quad h_i'' = \begin{cases} \frac{1}{h_i} & : h_i \neq 0, \\ 0 & : h_i = 0. \end{cases} \qquad (14)$$

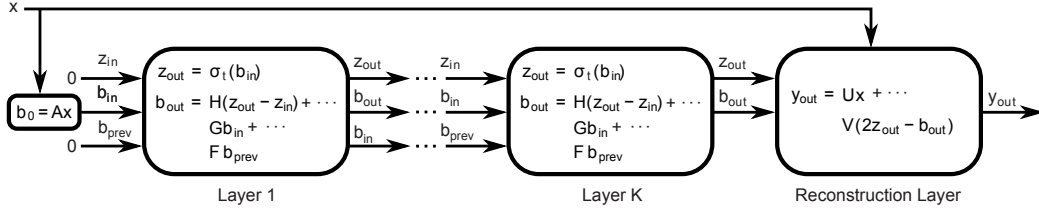

Figure 1: ADMM neural network encoder. The network comprises $K$ identical layers parameterized by the matrices $\mathbf{A}$ and $\mathbf{B}$ and the threshold vector $\mathbf{t}$, and one output layer parameterized by the matrices $\mathbf{U}$ and $\mathbf{V}$. The initial values of the learned parameters are given by ADMM (see Algorithm 1) according to $\mathbf{U} = (\mathbf{M}_2^T\mathbf{M}_2 + \rho\boldsymbol{\Omega}^T\boldsymbol{\Omega} + \lambda_2\mathbf{I})^{-1}\mathbf{M}_2^T\mathbf{M}_1$, $\mathbf{V} = \rho(\mathbf{M}_2^T\mathbf{M}_2 + \rho\boldsymbol{\Omega}^T\boldsymbol{\Omega} + \lambda_2\mathbf{I})^{-1}\boldsymbol{\Omega}^T$, $\mathbf{A} = \boldsymbol{\Omega}\mathbf{U}$, $\mathbf{H} = 2\boldsymbol{\Omega}\mathbf{V} - I$, $\mathbf{G} = 2I - \boldsymbol{\Omega}\mathbf{V}$, $\mathbf{F} = \boldsymbol{\Omega}\mathbf{V} - I$, and $\mathbf{t} = \frac{\lambda_1}{\rho}\mathbf{1}$.

**Proposition 1.** *The functional* $\mathbf{y}^* = \mathbf{\hat{y}}_\Theta^*(\mathbf{x})$ *in* (1) *is almost everywhere differentiable for* $\lambda_2 > 0$, *and its gradients satisfy*

$$\nabla_{\boldsymbol{\Omega}}\ell(\mathbf{y}^*) = -\lambda_1\text{sign}(\boldsymbol{\Omega}\mathbf{y}^*)\boldsymbol{\beta}^T - \mathbf{P}_{\Lambda^c}\boldsymbol{\Omega}(\mathbf{\tilde{y}}^*\boldsymbol{\beta}^T + \tilde{\boldsymbol{\beta}}\mathbf{y}^{*T}),$$
$$\nabla_{\mathbf{M}_1}\ell(\mathbf{y}^*) = \mathbf{M}_2(\mathbf{y}^*\boldsymbol{\beta}^T + \boldsymbol{\beta}\mathbf{y}^{*T}),$$

*where the vectors* $\boldsymbol{\beta}, \tilde{\boldsymbol{\beta}}$ *and* $\mathbf{\tilde{y}}$ *in* $\mathbb{R}^k$ *are defined as* $\boldsymbol{\beta} = \mathbf{Q}\nabla_{\mathbf{y}^*}\ell(\mathbf{x},\Theta)$, $\tilde{\boldsymbol{\beta}} = \mathbf{\tilde{Q}}\nabla_{\mathbf{y}^*}\ell(\mathbf{x},\Theta)$, *and* $\mathbf{\tilde{y}}^* = \mathbf{\tilde{Q}}(\mathbf{M}_2^T\mathbf{M}_1\mathbf{x} - \lambda_1\boldsymbol{\Omega}^T\text{sign}(\mathbf{z}^*))$, *with* $\mathbf{Q}$ *and* $\mathbf{\tilde{Q}}$ *given by* (14) *and* (15) *respectively.*

In addition to being a useful analytic tool, the relationship between (1) and its relaxed version (8) also has practical implications. Obtaining the exact gradients given in Proposition 1 requires computing the eigendecomposition of $\mathbf{C}$, which is in general computationally expensive. In practice, we approximate the gradients using the expressions in (12) and (13) with a fixed sufficiently large value of $t$. The supervised model learning framework can be straightforwardly specialized to the shift-invariant case, in which filters $\boldsymbol{\gamma}_i$ in (2) are learned instead of a full matrix $\boldsymbol{\Omega}$. The gradients of $\ell$ with respect to the filter coefficients are obtained using Proposition 1 and the chain rule.

## 4  Fast approximation

The discussed sparse models rely on an iterative optimization scheme such as ADMM, required to solve the pursuit problem (1). This has relatively high computational complexity and latency, which is furthermore data-dependent. ADMM typically requires hundreds or thousands of iterations to converge, greatly depending on the problem and the input. While the classical optimization theory provides worst-case (data-independent) convergence rate bounds for many families of iterative algorithms, very little is known about their behavior on *specific* data, coming, e.g., from a distribution supported on a low-dimensional manifold – characteristics often exhibited by real data. The common practice of sparse modeling concentrates on creating sophisticated data models, and then relies on computational and analytic techniques that are totally agnostic of the data structure. Such a discrepancy hides a (possibly dramatic) potential of computational improvement [11].

From the perspective of the pursuit process, the minimization of (1) is merely a proxy to obtaining a highly non-linear map between the data vector $\mathbf{x}$ and the representation vector $\mathbf{y}$ (which can also be the "feature" vector $\boldsymbol{\Omega}\mathbf{D}\mathbf{y}$ or the reconstructed data vector $\mathbf{D}\mathbf{y}$, depending on the application). Adopting ADMM, such a map can be expressed by unrolling a sufficient number $K$ of iterations into a feed-forward network comprising $K$ (identical) layers depicted in Figure 1, where the parameters $\mathbf{A}, \mathbf{B}, \mathbf{U}, \mathbf{V}$, and $\mathbf{t}$, collectively denoted as $\Psi$, are prescribed by the ADMM iteration. Fixing $K$, we obtain a fixed-complexity and latency encoder $\mathbf{\hat{y}}_{K,\Psi}(\mathbf{x})$, parameterized by $\Psi$.

Note that for a sufficiently large $K$, $\mathbf{\hat{y}}_{K,\Psi}(\mathbf{x}) \approx \mathbf{y}^*(\mathbf{x})$, with the latter denoting the exact minimizer of (1) given the input $\mathbf{x}$. However, when complexity budget constraints require $K$ to be truncated at a small fixed number, the output of $\mathbf{\hat{y}}_{K,\Psi}$ is usually unsatisfactory, and the worst-case analysis provided by the classical optimization theory is of little use. However, within the family of functions $\{\mathbf{\hat{y}}_{K,\Psi} : \Psi\}$, there might exist better parameters for which $\mathbf{\hat{y}}$ performs better *on relevant input data*. Such parameters can be obtained via learning, as described in the sequel.

Similar ideas were first advocated by [11], who considered Lasso sparse synthesis models, and showed that by unrolling iterative shrinkage thresholding algorithms (ISTA) into a neural network,

and learning a new set of parameters, approximate solutions to the pursuit problem could be obtained at a fraction of the cost of the exact solution, if the inputs were restricted to data coming from a distribution similar to that used at training. This approach was later extended to more elaborated structured sparse and low-rank models, with applications in audio separation and denoising [20]. Here is the first attempt to extend it to sparse analysis and mixed analysis-synthesis models.

The learning of the fast encoder is performed by plugging it into the training problem (4) in place of the exact encoder. The minimization of a loss function $\ell(\Psi)$ with respect to $\Psi$ requires the computation of the (sub)gradients $d\ell(\mathbf{y})/d\Psi$, which is achieved by the back-propagation procedure (essentially, an iterated application of the chain rule). Back-propagation starts with differentiating $\ell(\Psi)$ with respect to the output of the last network layer, and propagating the (sub)gradients down to the input layer, multiplying them by the Jacobian matrices of the traversed layers. For completeness, we summarize the procedure in the supplementary materials. There is no principled way of choosing the number of layers $K$ and in practice this is done via cross-validation. In Section 5 we discuss the selection of $K$ for a particular example.

In the particular setting of a shift-invariant analysis model, the described neural network encoder assumes a structure resembling that of a convolutional network. The matrices $\mathbf{A}, \mathbf{B}, \mathbf{U}$, and $\mathbf{V}$ parameterizing the network in Figure 1 are replaced by a set of filter coefficients. The initial inverse kernels of the form $(\rho\boldsymbol{\Omega}^{\mathrm{T}}\boldsymbol{\Omega}+(1+\lambda_2)\mathbf{I})^{-1}$ prescribed by ADMM are approximated by finite-support filters, which are computed using a standard least squares procedure.

## 5   Experimental results and discussion

In what follows, we illustrate the proposed approaches on two experiments: single-image super-resolution (demonstrating a reconstruction problem), and polyphonic music transcription (demonstrating a classification problem). Additional figures are provided in the supplementary materials.

**Single-image super-resolution.**   Single-image super-resolution is an inverse problem in which a high-resolution image is reconstructed from its blurred and down-sampled version lacking the high-frequency details. Low-resolution images were created by blurring the original ones with an anti-aliasing filter, followed by down-sampling operator. In [25], it has been demonstrated that pre-filtering a high resolution image with a Gaussian kernel with $\sigma = 0.8s$ guarantees that the following $s \times s$ sub-sampling generates an almost aliasing-free low resolution image. This models very well practical image decimation schemes, since allowing a certain amount of aliasing improves the visual perception. Super-resolution consists in inverting both the blurring and sub-sampling together as a compound operator. Since the amount of aliasing is limited, a bi-cubic spline interpolation is more accurate than lower ordered interpolations for restoring the images to their original size. As shown in [26], up-sampling the low resolution image in this way, produces an image that is very close to the pre-filtered high resolution counterpart. Then, the problem reduces to deconvolution with a Gaussian kernel. In all our experiments we used the scaling factor $s = 2$. A shift-invariant analysis model was tested in three configurations: a TV prior created using horizontal and vertical derivative filters; a bank of $48$ $7 \times 7$ non-constant DCT filters (referred to henceforth as *A-DCT*); and a combination of the former two settings tuned using the proposed supervised scheme with the loss function (5). The training set consisted of random image patches from [24]. We also tested a convolutional neural network approximation of the third model, trained under similar conditions. Pursuit problem was solved using ADMM with $\rho = 1$, requiring about 100 iterations to converge. Table 1 reports the obtained PSNR results on seven standard images used in super-resolution experiments. Visual results are shown in the supplementary materials. We observe that on the average, the supervised model outperforms A-DCT and TV by $1 - 3$ dB PSNR. While performing slightly inferior to the exact supervised model, the neural network approximation is about ten times faster.

**Automatic polyphonic music transcription.**   The goal of automatic music transcription is to obtain a musical score from an input audio signal. This task is particularly difficult when the audio signal is polyphonic, i.e., contains multiple pitches present simultaneously. Like the majority of music and speech analysis techniques, music transcription typically operates on the magnitude of the audio time-frequency representation such as the short-time Fourier transform or constant-Q transform (CQT) [7] (adopted here). Given a spectral frame $\mathbf{x}$ at some time, the transcription problem consists of producing a binary label vector $\mathbf{p} \in \{-1, +1\}^k$, whose $i$-th element indicates the pres-

| method | mean ±std. dev. | man | woman | barbara | boats | lena | house | peppers |
|---|---|---|---|---|---|---|---|---|
| Bicubic | $29.51 \pm 4.39$ | 28.52 | 38.22 | 24.02 | 27.38 | 30.77 | 29.75 | 27.95 |
| TV | $29.04 \pm 3.51$ | 30.23 | 33.39 | 24.25 | 29.44 | 31.75 | 29.91 | 24.31 |
| A-DCT | $31.06 \pm 4.84$ | 29.85 | 40.23 | 24.32 | 28.89 | 32.72 | 31.68 | 29.71 |
| SI-ADMM | $32.03 \pm 4.84$ | 31.05 | 40.62 | 24.55 | 30.06 | 34.06 | 32.91 | 30.93 |
| SI-NN ($K = 10$) | $31.53 \pm 5.03$ | 30.42 | 40.99 | 24.53 | 29.12 | 33.58 | 31.82 | 30.21 |

Table 1: PSNR in dB of different image super-resolution methods: bicubic interpolation (Bicubic), shift-invariant analysis models with TV and DCT priors (TV and A-DCT), supervised shift-invariant analysis model (SI-ADMM), and its fast approximation with $K = 10$ layers (SI-NN).

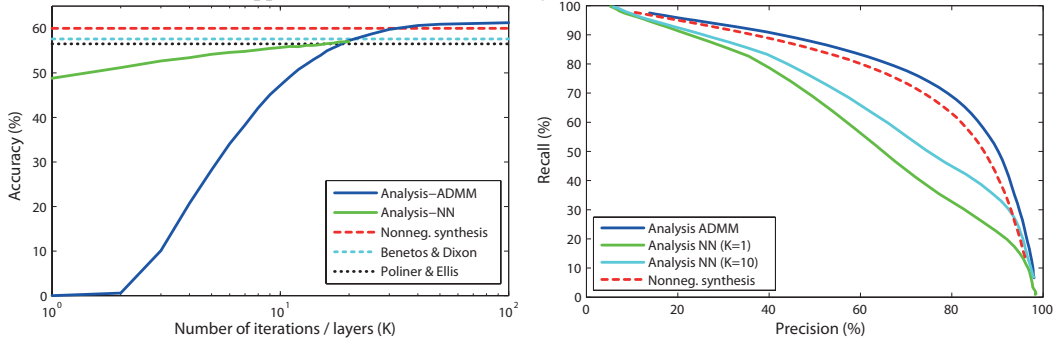

Figure 2: Left: Accuracy of the proposed analysis model (Analysis-ADMM) and its fast approximation (Analysis-NN) as the function of number of iterations or layers $K$. For reference, the accuracy of a non-negative synthesis model as well as two leading methods [3, 18] is shown. Right: Precision-recall curve.

ence ($+1$) or absence ($-1$) of the $i$-th pitch at that time. We use $k = 88$ corresponding to the span of the standard piano keyboard (MIDI pitches $21 - 108$).

We used an analysis model with a square dictionary $\mathbf{\Omega}$ and a square metric matrix $\mathbf{M}_1 = \mathbf{M}_2$ to produce the feature vector $\mathbf{z} = \mathbf{\Omega}\mathbf{y}$, which was then fed to a classifier of the form $\mathbf{p} = \text{sign}(\mathbf{W}\mathbf{z}+\mathbf{b})$. The parameters $\mathbf{\Omega}$, $\mathbf{M}_2$, $\mathbf{W}$, and $\mathbf{b}$ were trained using the logistic loss on the MAPS Disklavier dataset [10] containing examples of polyphonic piano recordings with time-aligned groundtruth. The testing was performed on another annotated real piano dataset from [18]. Transcription was performed frame-by-frame, and the output of the classifier was temporally filtered using a hidden Markov model proposed in [3]. For comparison, we show the performance of a supervised non-negative synthesis model and two leading methods [3, 18] evaluated in the same settings.

Performance was measured using the standard precision-recall curve depicted in Figure 2 (right); in addition we used accuracy measure $\text{Acc} = \text{TP}/(\text{FP} + \text{FN} + \text{TP})$, where TP (true positives) is the number of correctly predicted pitches, and FP (false positives) and FN (false negatives) are the number of pitches incorrectly transcribed as ON or OFF, respectively. This measure is frequently used in the music analysis literature [3, 18]. The supervised analysis model outperforms leading pitch transcription methods. Figure 2 (left) shows that replacing the exact ADMM solver by a fast approximation described in Section 4 achieves comparable performance, with significantly lower complexity. In this example, ten layers are enough for having a good representation and the improvement obtained by adding layers begins to be very marginal around this point.

**Conclusion.** We presented a bilevel optimization framework for the supervised learning of a superset of sparse analysis and synthesis models. We also showed that in applications requiring low complexity or latency, a fast approximation to the exact solution of the pursuit problem can be achieved by a feed-forward architecture derived from truncated ADMM. The obtained fast regressor can be initialized with the model parameters trained through the supervised bilevel framework, and tuned similarly to the training and adaptation of neural networks. We observed that the structure of the network becomes essentially a convolutional network in the case of shift-invariant models. The generative setting of the proposed approaches was demonstrated on an image restoration experiment, while the discriminative setting was tested in a polyphonic piano transcription experiment. In the former we obtained a very good and fast solution while in the latter the results comparable or superior to the state-of-the-art.

## Footnotes

*Work partially supported by ARO, BSF, NGA, ONR, NSF, NSSEFF, and Israel-Us Binational.

# References

[1] M. Aharon, M. Elad, and A. Bruckstein. $k$-SVD: an algorithm for designing overcomplete dictionaries for sparse representation. *IEEE Trans. Sig. Proc.*, 54(11):4311–4322, 2006.

[2] A. Beck and M. Teboulle. A fast iterative shrinkage-thresholding algorithm for linear inverse problems. *SIAM J. Img. Sci.*, 2:183–202, March 2009.

[3] E. Benetos and S. Dixon. Multiple-instrument polyphonic music transcription using a convolutive probabilistic model. In *Sound and Music Computing Conference*, pages 19–24, 2011.

[4] D.P. Bertsekas. Nonlinear programming. 1999.

[5] H. Bischof, Y. Chen, and T. Pock. Learning l1-based analysis and synthesis sparsity priors using bi-level optimization. *NIPS workshop*, 2012.

[6] M. M. Bronstein, A. M. Bronstein, M. Zibulevsky, and Y. Y. Zeevi. Blind deconvolution of images using optimal sparse representations. *IEEE Trans. Im. Proc.*, 14(6):726–736, 2005.

[7] J. C. Brown. Calculation of a constant Q spectral transform. *The Journal of the Acoustical Society of America*, 89:425, 1991.

[8] B. Colson, P. Marcotte, and G. Savard. An overview of bilevel optimization. *Annals of operations research*, 153(1):235–256, 2007.

[9] M. Elad and M. Aharon. Image denoising via sparse and redundant representations over learned dictionaries. *IEEE Trans. on Im. Proc.*, 54(12):3736–3745, 2006.

[10] V. Emiya, R. Badeau, and B. David. Multipitch estimation of piano sounds using a new probabilistic spectral smoothness principle. *IEEE Trans. Audio, Speech, and Language Proc.*, 18(6):1643–1654, 2010.

[11] K. Gregor and Y. LeCun. Learning fast approximations of sparse coding. In *ICML*, pages 399–406, 2010.

[12] J. Mairal, F. Bach, and J. Ponce. Task-driven dictionary learning. *IEEE Trans. PAMI*, 34(4):791–804, 2012.

[13] J. Mairal, M. Elad, and G. Sapiro. Sparse representation for color image restoration. *IEEE Trans. on Im. Proc.*, 17(1):53–69, 2008.

[14] S. Mallat. *A Wavelet Tour of Signal Processing, Second Edition*. Academic Press, 1999.

[15] Y. Nesterov. Gradient methods for minimizing composite objective function. In *CORE*. Catholic University of Louvain, Louvain-la-Neuve, Belgium, 2007.

[16] B.A. Olshausen and D. J. Field. Emergence of simple-cell receptive field properties by learning a sparse code for natural images. *Nature*, 381(6583):607–609, 1996.

[17] G. Peyré and J. Fadili. Learning analysis sparsity priors. *SAMPTA'11*, 2011.

[18] G. E. Poliner and D. Ellis. A discriminative model for polyphonic piano transcription. *EURASIP J. Adv. in Sig. Proc.*, 2007, 2006.

[19] L.I. Rudin, S. Osher, and E. Fatemi. Nonlinear total variation-based noise removal algorithms. *Physica D*, 60(1-4):259–268, 1992.

[20] P. Sprechmann, A. M. Bronstein, and G. Sapiro. Learning efficient sparse and low rank models. *arXiv preprint arXiv:1212.3631*, 2012.

[21] R. Tibshirani. Regression shrinkage and selection via the LASSO. *J. Royal Stat. Society: Series B*, 58(1):267–288, 1996.

[22] Ryan Joseph Tibshirani. *The solution path of the generalized lasso*. Stanford University, 2011.

[23] S. Vaiter, G. Peyre, C. Dossal, and J. Fadili. Robust sparse analysis regularization. *Information Theory, IEEE Transactions on*, 59(4):2001–2016, 2013.

[24] J. Yang, John W., T. Huang, and Y. Ma. Image super-resolution as sparse representation of raw image patches. In *Proc. CVPR*, pages 1–8. IEEE, 2008.

[25] G. Yu and J.-M. Morel. On the consistency of the SIFT method. *Inverse problems and Imaging*, 2009.

[26] G. Yu, G. Sapiro, and S. Mallat. Solving inverse problems with piecewise linear estimators: from gaussian mixture models to structured sparsity. *IEEE Trans. Im. Proc.*, 21(5):2481–2499, 2012.

